# On the Power of Neural Networks for Solving Hard Problems

Jehoshua Bruck
Joseph W. Goodman
Information Systems Laboratory
Department of Electrical Engineering
Stanford University
Stanford, CA 94305

## Abstract

This paper deals with a neural network model in which each neuron performs a threshold logic function. An important property of the model is that it always converges to a stable state when operating in a serial mode [2,5]. This property is the basis of the potential applications of the model such as associative memory devices and combinatorial optimization [3,6].

One of the motivations for use of the model for solving hard combinatorial problems is the fact that it can be implemented by optical devices and thus operate at a higher speed than conventional electronics.

The main theme in this work is to investigate the power of the model for solving NP-hard problems [4,8], and to understand the relation between speed of operation and the size of a neural network. In particular, it will be shown that for any NP-hard problem the existence of a polynomial size network that solves it implies that NP=co-NP. Also, for Traveling Salesman Problem (TSP), even a polynomial size network that gets an $\epsilon$-approximate solution does not exist unless P=NP.

The above results are of great practical interest, because right now it is possible to build neural networks which will operate fast but are limited in the number of neurons.

# 1 Background

The neural network model is a discrete time system that can be represented by a weighted and undirected graph. There is a weight attached to each edge of the graph and a threshold value attached to each node (neuron) of the graph.

The *order* of the network is the number of nodes in the corresponding graph. Let $N$ be a neural network of order $n$; then $N$ is uniquely defined by $(W, T)$ where:

- $W$ is an $n \times n$ symmetric matrix, $W_{ij}$ is equal to the weight attached to edge $(i, j)$.

- $T$ is a vector of dimension $n$, $T_i$ denotes the threshold attached to node $i$.

Every node (neuron) can be in one of two possible states, either 1 or -1. The state of node $i$ at time $t$ is denoted by $V_i(t)$. The *state* of the neural network at time $t$ is the vector $V(t)$.

The next state of a node is computed by:

$$V_i(t+1) = sgn(H_i(t)) = \begin{cases} 1 & \text{if } H_i(t) \geq 0 \\ -1 & \text{otherwise} \end{cases} \tag{1}$$

where

$$H_i(t) = \sum_{j=1}^{n} W_{ji} V_j(t) - T_i$$

The next state of the network, i.e. $V(t+1)$, is computed from the current state by performing the evaluation (1) at a subset of the nodes of the network, to be denoted by $S$. The modes of operation are determined by the method by which the set $S$ is selected in each time interval. If the computation is performed at a single node in any time interval, i.e. $| S |= 1$, then we will say that the network is operating in a *serial* mode; if $| S |= n$ then we will say that that the network is operating in a *fully parallel* mode. All the other cases, i.e. $1 <| S |< n$ will be called *parallel* modes of operation. The set $S$ can be chosen at random or according to some deterministic rule.

A state $V(t)$ is called *stable* iff $V(t) = sgn(WV(t) - T)$, i.e. there is no change in the state of the network no matter what the mode of operation is. One of the most important properties of the model is the fact that it always converges to a stable state while operating in a serial mode. The main idea in the proof of the convergence property is to define a so called *energy function* and to show that this energy function is nondecreasing when the state of the network changes. The energy function is:

$$E(t) = V^T(t) W V(t) - 2V^T(t) T \tag{2}$$

An important note is that originally the energy function was defined such that it is nonincreasing [5]; we changed it such that it will comply with some known graph problems (e.g. Min Cut).

A neural network will always get to a stable state which corresponds to a local maximum in the energy function. This suggests the use of the network as a

device for performing a local search algorithm for finding a maximal value of the energy function [6]. Thus, the network will perform a local search by operating in a random and serial mode. It is also known [2,9] that maximization of $E$ associated with a given network $N$ in which $T = 0$ is equivalent to finding the Minimum Cut in $N$. Actually, many hard problems can be formulated as maximization of a quadratic form (e.g. TSP [6]) and thus can be mapped to a neural network.

# 2   The Main Results

The set of stable states is the set of possible final solutions that one will get using the above approach. These final solutions correspond to local maxima of the energy function but do not necessarily correspond to global optima of the corresponding problem. The main question is: suppose we allow the network to operate for a very long time until it converges; can we do better than just getting some local optimum? i.e., is it possible to design a network which will always find the exact solution (or some guaranteed approximation) of the problem?

**Definition:** Let $X$ be an instance of problem. Then $| X |$ denotes the size of $X$, that is, the number of bits required to represent $X$. For example, for $X$ being an instance of TSP, $| X |$ is the number of bits needed to represent the matrix of the distances between cities.

**Definition:** Let $N$ be a neural network. Then $| N |$ denotes the size of the network $N$. Namely, the number of bits needed to represent $W$ and $T$.

Let us start by defining the desired setup for using the neural network as a model for solving hard problems.

Consider an optimization problem $L$, we would like to have for every instance $X$ of $L$ a neural network $N_X$ with the following properties:

- Every local maximum of the energy function associated with $N_X$ corresponds to a global optimum of $X$.

- The network $N_X$ is small, that is, $| N_X |$ is bounded by some polynomial in $| X |$.

Moreover, we would like to have an algorithm, to be denoted by $A_L$, which given an instance $X \in L$, generates the description for $N_X$ in polynomial (in $| X |$) time.

Now, we will define the desired setup for using the neural network as a model for finding approximate solutions for hard problems.

**Definition:** Let $E_{glo}$ be the global maximum of the energy function. Let $E_{loc}$

be a local maximum of the energy function. We will say that a local maximum is an $\epsilon$-approximate of the global iff:

$$\frac{E_{glo} - E_{loc}}{E_{glo}} \leq \epsilon$$

The setup for finding approximate solutions is similar to the one for finding exact solutions. For $\epsilon \geq 0$ being some fixed number. We would like to have a network $N_{X_\epsilon}$ in which every local maximum is an $\epsilon$-approximate of the global and that the global corresponds to an optimum of $X$. The network $N_{X_\epsilon}$ should be small, namely, $\mid N_{X_\epsilon} \mid$ should be bounded by a polynomial in $\mid X \mid$. Also, we would like to have an algorithm $A_{L_\epsilon}$, such that, given an instance $X \in L$, it generates the description for $N_{X_\epsilon}$ in polynomial (in $\mid X \mid$) time.

Note that in both the exact case and the approximate case we do not put any restriction on the time it takes the network to converge to a solution (it can be exponential).

At this point the reader should convince himself that the above description is what he imagined as the setup for using the neural network model for solving hard problems, because that is what the following definition is about.

**Definition:** We will say that a neural network for solving (or finding an $\epsilon$-approximation of) a problem $L$ exists if the algorithm $A_L$ (or $A_{L_\epsilon}$) which generates the description of $N_X$ (or $N_{X_\epsilon}$) exists.

The main results in the paper are summarized by the following two propositions. The first one deals with exact solutions of NP-hard problems while the second deals with approximate solutions to TSP.

**Proposition 1** *Let $L$ be an NP-hard problem. Then the existence of a neural network for solving $L$ implies that NP = co-NP.*

**Proposition 2** *Let $\epsilon \geq 0$ be some fixed number. The existence of a neural network for finding an $\epsilon$-approximate solution to TSP implies that P=NP.*

Both (P=NP) and (NP=co-NP) are believed to be false statements, hence, we can not use the model in the way we imagine.

The key observation for proving the above propositions is the fact that a single iteration in a neural network takes time which is bounded by a polynomial in the size of the instance of the corresponding problem. The proofs of the above two propositions follow directly from known results in complexity theory and should not be considered as new results in complexity theory.

# 3  The Proofs

**Proof of Proposition 1:** The proof follows from the definition of the classes NP and co-NP, and Lemma 1. The definitions and the lemma appear in Chapters 15 and 16 in [8] and also in Chapters 2 and 7 in [4].

**Lemma 1** *If the complement of an NP-complete problem is in NP, then NP=co-NP.*

Let $L$ be an NP-hard problem. Suppose there exists a neural network that solves $L$. Let $\hat{L}$ be an NP-complete problem. By definition, $\hat{L}$ can be polynomialy reduced to $L$. Thus, for every instance $X \in \hat{L}$, we have a neural network such that from any of its global maxima we can efficiently recognize whether $X$ is a 'yes' or a 'no' instance of $\hat{L}$.

We claim that we have a nondeterministic polynomial time algorithm to decide that a given instance $X \in \hat{L}$ is a 'no' instance. Here is how we do it: for $X \in \hat{L}$ we construct the neural network that solves it by using the reduction to $L$. We then check every state of the network to see if it is a local maximum (that is done in polynomial time). In case it is a local maximum, we check if the instance is a 'yes' or a 'no' instance (this is also done in polynomial time).

Thus, we have a nondeterministic polynomial time algorithm to recognize any 'no' instance of $\hat{L}$. Thus, the complement of the problem $\hat{L}$ is in NP. But $\hat{L}$ is an NP-complete problem, hence, from Lemma 1 it follows that NP=co-NP. □

**Proof of Proposition 2:** The result is a corollary of the results in [7], the reader can refer to it for a more complete presentation.

The proof uses the fact that the Restricted Hamiltonian Circuit (RHC) is an NP-complete problem.

**Definiton of RHC:** Given a graph $G = (V, E)$ and a Hamiltonian path in $G$. The question is whether there is a Hamiltonian circuit in $G$?

It is proven in [7] that RHC is NP-complete.

Suppose there exists a polynomial size neural network for finding an $\epsilon$-approximate solution to TSP. Then it can be shown that an instance $X \in RHC$ can be reduced to an instance $\hat{X} \in TSP$, such that in the network $N_{\hat{X}_\epsilon}$ the following holds: if the Hamiltonian path that is given in $X$ corresponds to a local maximum in $N_{\hat{X}_\epsilon}$ then X is a 'no' instance; else, if it does not correspond to a local maximum in $N_{\hat{X}_\epsilon}$ then $X$ is a 'yes' instance. Note that we can check for locality in polynomial time.

Hence, the existence of $N_{\hat{X}_\epsilon}$ for all $\hat{X} \in TSP$ implies that we have a polynomial time algorithm for RHC. □

# 4   Concluding Remarks

1. In Proposition 1 we let $|W|$ and $|T|$ be arbitrary but bounded by a polynomial in the size of a given instance of a problem. If we assume that $|W|$ and $|T|$ are fixed for all instances then a similar result to Proposition 1 can be proved without using complexity theory; this result appears in [1].

2. The network which corresponds to TSP, as suggested in [6], can not solve the TSP with guaranteed quality. However, one should note that all the analysis in this paper is a worst case type of analysis. So, it might be that there exist networks that have good behavior on the average.

3. Proposition 1 is general to all NP-hard problems while Proposition 2 is specific to TSP. Both propositions hold for any type of networks in which an iteration takes polynomial time.

4. Clearly, every network has an algorithm which is equivalent to it, but an algorithm does not necessarily have a corresponding network. Thus, if we do not know of an algorithmic solution to a problem we also will not be able to find a network which solves the problem. If one believes that the neural network model is a good model (e.g. it is amenable to implementation with optics), one should develop techniques to program the network to perform an algorithm that is known to have some guaranteed good behavior.

**Acknowledgement:** Support of the U.S. Air Force Office of Scientific Research is gratefully acknowledged.

# References

[1] Y. Abu Mostafa, *Neural Networks for Computing?* in Neural Networks for Computing, edited by J. Denker (AIP Conference Proceedings no. 151, 1986).

[2] J. Bruck and J. Sanz, *A Study on Neural Networks*, IBM Tech Rep, RJ 5403, 1986. To appear in International Journal of Intelligent Systems, 1988.

[3] J. Bruck and J. W. Goodman, *A Generalized Convergence Theorem for Neural Networks and its Applications in Combinatorial Optimization*, IEEE First ICNN, San-Diego, June 1987.

[4] M. R. Garey and D. S. Johnson, *Computers and Intractability: A Guide to the Theory of NP-Completeness*, W. H. Freeman and Company, 1979.

[5] J. J. Hopfield, *Neural Networks and Physical Systems with Emergent Collective Computational Abilities*, Proc. Nat. Acad. Sci. . USA, Vol. 79, pp. 2554-2558, 1982.

[6] J. J. Hopfield and D. W. Tank, *Neural Computations of Decisions in Optimization Problems*, Biol. Cybern. 52, pp. 141-152, 1985.

[7] C. H. Papadimitriou and K. Steiglitz, *On the Complexity of Local Search for the Traveling Salesman Problem*, SIAM J. on Comp., Vol. 6, No. 1, pp. 76-83, 1977.

[8] C. H. Papadimitriou and K. Steiglitz, *Combinatorial Optimization: Algorithms and Complexity*, Prentice-Hall, Inc., 1982.

[9] J. C. Picard and H. D. Ratliff, *Minimum Cuts and Related Problems*, Networks, Vol 5, pp. 357-370, 1974.
